# Analytical solution of spike-timing dependent plasticity based on synaptic biophysics

**Bernd Porr, Ausra Saudargiene and Florentin Wörgötter**
Computational Neuroscience
Psychology
University of Stirling
FK9 4LR Stirling, UK
`{Bernd.Porr,ausra,worgott}@cn.stir.ac.uk`

## Abstract

Spike timing plasticity (STDP) is a special form of synaptic plasticity where the relative timing of post- and presynaptic activity determines the change of the synaptic weight. On the postsynaptic side, active back-propagating spikes in dendrites seem to play a crucial role in the induction of spike timing dependent plasticity. We argue that postsynaptically the temporal change of the membrane potential determines the weight change. Coming from the presynaptic side induction of STDP is closely related to the activation of NMDA channels. Therefore, we will calculate analytically the change of the synaptic weight by correlating the derivative of the membrane potential with the activity of the NMDA channel. Thus, for this calculation we utilise biophysical variables of the physiological cell. The final result shows a weight change curve which conforms with measurements from biology. The positive part of the weight change curve is determined by the NMDA activation. The negative part of the weight change curve is determined by the membrane potential change. Therefore, the weight change curve should change its shape depending on the distance from the soma of the postsynaptic cell. We find temporally asymmetric weight change close to the soma and temporally symmetric weight change in the distal dendrite.

## 1 Introduction

Donald Hebb [1] postulated half a century ago that the change of synaptic strength depends on the correlation of pre- and postsynaptic activity: cells which fire together wire together. Here we want to concentrate on a special form of correlation based learning, namely, spike timing dependent plasticity (STDP, [2, 3]). STDP is asymmetrical in time: Weights grow if the pre-synaptic event precedes the postsynaptic event. This phenomenon is called long-term potentiation (LTP). Weights shrink when the temporal order is reversed. This is called long-term depression (LTD).

Correlations between pre- and postsynaptic activity can take place at different locations of the cell. Here we will focus on the dendrite of the cell (see Fig. 1). The dendrite has attracted interest recently because of its ability to propagate spikes back from the soma

of the cell into its distal regions. Such spikes are called backpropagating spikes. The transmission is active which guarantees that the spikes can reach even the distal regions of the dendrite [4]. Backpropagating spikes have been suggested to be the driving force for STDP in the dendrite [5]. On the presynaptic side the main contribution to STDP comes from $Ca2+$ flow through the NMDA channels [6].

The goal of this study is to derive an analytical solution for STDP on the basis of the biophysical properties of the NMDA channel and the cell membrane. We will show that mainly the timing of the backpropagating spike determines the shape of the learning curve. With fast decaying backpropagating spikes we obtain STDP while with slow decaying backpropagating spikes we approximate temporally symmetric Hebbian learning.

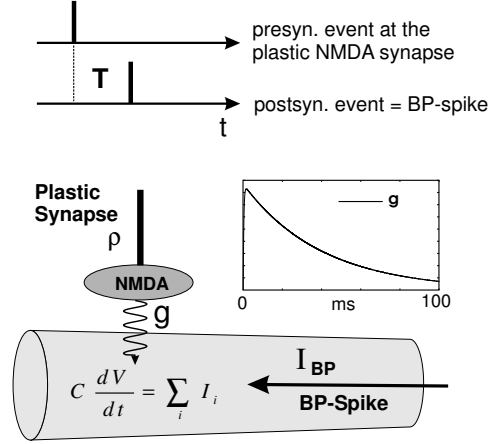

Figure 1: Schematic diagram of the model setup. The inset shows the time course of an NMDA response as modelled by Eq. 2.

## 2   The Model

The goal is to define a weight change rule which correlates the dynamics of an NMDA channel with a variable which is linked to the dynamics of a backpropagating spike. The precise biophysical mechanisms of STDP are still to a large degree unresolved. It is, however, known that high levels of $Ca^{2+}$ concentration resulting from $Ca^{2+}$ influx mainly through NMDA-channels will lead to LTP, while lower levels will lead to LTD. Several biophysically more realistic models for STDP were recently designed which rely on this mechanisms [7, 8, 9]. Recent physiological results (reviewed in detail in [10]), however suggest that not only the $Ca^{2+}$ concentration but maybe more importantly the *change* of the $Ca^{2+}$ concentration determines if LTP or LTD is observed. This clearly suggests that a differential term should be included in the learning rule, when trying to model STDP. On theoretical grounds such a suggestion has also been made by several authors [11] who discussed that the abstract STDP models [12] are related to the much older model class of differential Hebbian learning rules [13]. In our model we assume that the $Ca^{2+}$ concentration and the membrane potential are highly correlated. Consequently, our learning rule utilises the derivative of the membrane potential for the postsynaptic activity.

After having identified the postsynaptic part of the weight change rule we have to define the presynaptic part. This shall be the conductance function of the NMDA channel [6].

The conventional membrane equation reads:

$$C\frac{dv(t)}{dt} = \rho\, g(t)[E - v(t)] + i_{BP}(t) + \frac{V_{rest} - v(t)}{R}, \qquad (1)$$

where $v$ is the membrane potential, $\rho$ the synaptic weight of the NMDA-channel and $g$, $E$ are its conductance and equilibrium potential, respectively. The current, which a BP-spike elicits, is given by $i_{BP}$ and the last term represents the passive repolarisation property of the membrane towards its resting potential $V_{rest} = -70\ mV$. We set the membrane capacitance $C = 50\ pF$ and the membrane resistance to $R = 100\ M\Omega$. $E$ is set to zero. The NMDA channel has the following equation:

$$g(t) = \bar{g}\frac{e^{-b_1 t} - e^{-a_1 t}}{[a_1 - b_1][1 + \kappa e^{-\gamma V(t)}]} \qquad (2)$$

For simpler notation, in general we use inverse time-constants $a_1 = \tau_a^{-1}$, $b_1 = \tau_b^{-1}$, etc. In addition, the term $a_1 - b_1$ in the denominator is required for later easier integration in the Laplace domain. Thus, we adjust for this by defining $\bar{g} = 12\ mS/ms$ which represents the peak conductance ($4\ nS$) multiplied by $b_1 - a_1$. The other parameters were: $a_1 = 3.0/ms$, $b_1 = 0.025/ms$, $\gamma = 0.06/mV$. Since we will not vary the $Mg^{2+}$ concentration we have already abbreviated: $\kappa = \eta[Mg^{2+}]$, $\eta = 0.33/mM$, $[Mg^{2+}] = 1\ mM$ [14].

The synaptic weight of the NMDA channel is changed by correlating the conductance of this NMDA channel with the change (derivative) of the membrane potential:

$$\frac{d}{dt}\rho = g(t)v'(t) \qquad (3)$$

To describe the weight change, we wish to solve:

$$\Delta\rho(T) = \int_0^\infty g(T + \tau)v'(\tau)d\tau, \qquad (4)$$

where $T$ is the temporal shift between the presynaptic activity and the postsynaptic activity. The shift $T > 0$ means that the backpropagating spike follows after the trigger of the NMDA channel. The shift $T < 0$ means that the temporal sequence of the pre- and postsynaptic events is reversed.

To solve Eq. 4 we have to simplify it, however, without loosing biophysical realism. In this paper we are interested in different shapes of backpropagating spikes. The underlying mechanisms which establish backpropagating spikes will not be addressed here. The backpropagating spike shall be simply modelled as a potential change in the dendrite and its shape is determined by its amplitude, its rise time and its decay time.

First we observe that the influence of a single (or even a few) NMDA-channels on the membrane potential can be neglected in comparison to a BP-spike[1], which, due to active processes, leads to a depolarisation of often more than $50\ mV$ even at distal dendrites because of active processes [15]. Thus, we can assume that the dynamics of the membrane potential is established by the backpropagating spike and the resting potential $V_{rest}$:

$$C\frac{dv(t)}{dt} = i_{BP}(t) + \frac{V_{rest} - v(t)}{R} \qquad (5)$$

This equation can be further simplified. Next we assume that the second passive repolarisation term can also be absorbed into $i_{BP}$, thus resulting to $i_{total}(t) = i_{BP}(t) + \frac{V_{rest} - v(t)}{R}$. To this end we model $i_{total}$ as a derivative of a band-pass filter function:

$$i_{total}(t) = \bar{i}_{total}\frac{a_2 e^{-a_2 t} - b_2 e^{-b_2 t}}{a_2 - b_2} \qquad (6)$$

where $\bar{i}_{total}$ is the current amplitude. This filter function causes first an influx of charges into the dendrite and then again an outflux of charges. The time constants $a_2$ and $b_2$ determine the timing of the current flow and therefore the rise and decay time. The total charge flux is zero so that the resting potential is reestablished after a backpropagating spike.

In this way the active de- and repolarising properties of a BP-spike can be combined with the passive properties of the membrane, in practise by a curve fitting procedure which yields $a_2, b_2$. As a result we find that the membrane equation in our case reduces to:

$$C\frac{dv(t)}{dt} = i_{total}(t) \tag{7}$$

We receive the resulting membrane potential simply by integrating Eq. 6:

$$v(t) = \frac{\bar{i}_{total}}{C}\frac{e^{-b_2 t} - e^{-a_2 t}}{a_2 - b_2} \tag{8}$$

Note the sign inversion between $v$ (Eq. 8) and $i$ (Eq. 6, the one being the derivative of the other.

The NMDA conductance $g$ is more complex, because the membrane potential enters the denominator in Eq. 2. To simplify we perform a Taylor expansion around $v = 0\,mV$. We expand around $0\,mV$ and not around the resting potential. There are two reasons. First, we are interested in the *open* NMDA channel. This is the case for voltages towards $0\,mV$. Second, the NMDA channel has a strong non-linearity around the resting potential. Towards $0\,mV$, however, the NMDA channel has a linear voltage/current curve. Therefore it makes sense to expand around $0\,mV$.

The NMDA conductance can now be written as:

$$g(t) = \bar{g}\frac{e^{-b_1 t} - e^{-a_1 t}}{a_1 - b_1} \cdot \left(\frac{1}{\kappa + 1} + \frac{\gamma \kappa v(t)}{(\kappa + 1)^2} + \ldots\right) \tag{9}$$

and finally the potential $v(t)$ (Eq. 8) can be inserted:

$$g(t) = \bar{g}\frac{e^{-b_1 t} - e^{-a_1 t}}{a_1 - b_1} \cdot \tag{10}$$

$$\left(\frac{1}{\kappa + 1} + \frac{\bar{i}_{total}\gamma\kappa e^{-b_2 t}}{C(\kappa + 1)^2(a_2 - b_2)} - \frac{\bar{i}_{total}\gamma\kappa e^{-a_2 t}}{C(\kappa + 1)^2(a_2 - b_2)} + \ldots\right) \tag{11}$$

terminating the Taylor series after the second term this leads to three contributions to the conductance:

$$g(t) = \underbrace{\frac{\bar{g}}{\kappa + 1}\frac{e^{-b_1 t} - e^{-a_1 t}}{a_1 - b_1}}_{g^{(0)}} \tag{12}$$

$$\underbrace{-\frac{\bar{g}\bar{i}_{total}\gamma\kappa}{(\kappa + 1)^2 C}\frac{e^{-(b_1 + a_2)t} - e^{-(a_1 + a_2)t}}{(a_1 - b_1)(a_2 - b_2)}}_{g^{(1a)}} \tag{13}$$

$$\underbrace{+\frac{\bar{g}\bar{i}_{total}\gamma\kappa}{(\kappa + 1)^2 C}\frac{e^{-(b_1 + b_2)t} - e^{-(a_1 + b_2)t}}{(a_1 - b_1)(a_2 - b_2)}}_{g^{(1b)}} \tag{14}$$

To perform the correlation in Eq. 4 we transform the required terms into the Laplace domain getting:

$$g^{(0,1a,1b)}(t) = k\frac{e^{-\beta t} - e^{-\alpha t}}{\alpha - \beta} \quad \leftrightarrow \quad G^{(0,1a,1b)}(s) = k\frac{1}{(s + \alpha)(s + \beta)} \tag{15}$$

$$i_{total}(t) = \bar{i}_{total}\frac{a_2 e^{-a_2 t} - b_2 e^{-b_2 t}}{a_2 - b_2} \quad \leftrightarrow \quad I_{total}(s) = \bar{i}_{total}\frac{s}{(s + a_2)(s + b_2)} \tag{16}$$

where $\alpha$ and $\beta$ take the coefficient values from the exponential terms in $g^{(0)}, g^{(1a)}, g^{(1b)}$, respectively and $k$ are the corresponding multiplicative factors[2].

A correlation in the Laplace domain is expressed by Plancherel's theorem [16]:

$$\Delta\rho = \frac{1}{2\pi} \left( \int_{-\infty}^{+\infty} G^{(0)}(-\imath\omega)e^{-\imath\omega T}I_t(\imath\omega)d\omega \right. \tag{17}$$

$$- \int_{-\infty}^{+\infty} G^{(1a)}(-\imath\omega)e^{-\imath\omega T}I_t(\imath\omega)d\omega \tag{18}$$

$$\left. + \int_{-\infty}^{+\infty} G^{(1b)}(-\imath\omega)e^{-\imath\omega T}I_t(\imath\omega)d\omega \right) \tag{19}$$

The solution is calculated with the method of residuals which leads to a split of the result into $T \geq 0$ and $T < 0$ and we get:

For $T \geq 0$:

$$\Delta\rho(T) = \frac{\bar{g}\bar{i}_{total}}{(\kappa+1)C} \left[ \frac{b_1 e^{-b_1 T}}{B_+^{(0)}} - \frac{a_1 e^{-a_1 T}}{A_+^{(0)}} \right. \tag{20}$$

$$- \frac{\gamma\kappa\bar{i}_{total}}{(\kappa+1)(a_2-b_2)C} \left( \frac{(b_1+a_2)e^{-(b_1+a_2)T}}{B_+^{(1)}} - \frac{(a_1+a_2)e^{-(a_1+a_2)T}}{A_+^{(1)}} \right) \tag{21}$$

$$\left. + \frac{\gamma\kappa\bar{i}_{total}}{(\kappa+1)(a_2-b_2)C} \left( \frac{(b_1+b_2)e^{-(b_1+b_2)T}}{B_+^{(1)}} - \frac{(a_1+b_2)e^{-(a_1+b_2)T}}{A_+^{(1)}} \right) \right] \tag{22}$$

with $A_+^{(0)} = (a_1-b_1)(a_1+a_2)(a_1+b_2)$, $A_+^{(1)} = (a_1-b_1)(a_1+2a_2)(a_1+a_2+b_2)$, $B_+^{(0)} = (a_1-b_1)(b_1+b_2)(a_2+b_1)$, $B_+^{(1)} = (a_1-b_1)(2a_2+b_1)(a_2+b_1+b_2)$.

For $T < 0$:

$$\Delta\rho(T) = \frac{\bar{g}\bar{i}_{total}}{(\kappa+1)C} \left[ \frac{a_2 e^{a_2 T}}{A_-^{(0)}} - \frac{b_2 e^{b_2 T}}{B_-^{(0)}} \right. \tag{23}$$

$$- \frac{\gamma\kappa\bar{i}_{total}}{(\kappa+1)(a_2-b_2)C} \left( \frac{a_2 e^{a_2 T}}{A_-^{(1a)}} - \frac{b_2 e^{b_2 T}}{B_-^{(1a)}} \right) \tag{24}$$

$$\left. + \frac{\gamma\kappa\bar{i}_{total}}{(\kappa+1)(a_2-b_2)C} \left( \frac{a_2 e^{a_2 T}}{A_-^{(1b)}} - \frac{b_2 e^{b_2 T}}{B_-^{(1b)}} \right) \right] \tag{25}$$

with $A_-^{(0)} = (a_2-b_2)(a_1+a_2)(a_2+b_1)$, $A_-^{(1a)} = (a_2-b_2)(a_1+2a_2)(2a_2+b_1)$, $A_-^{(1b)} = (a_2-b_2)(a_1+b_2+a_2)(a_2+b_1+b_2)$, $B_-^{(0)} = (a_2-b_2)(a_1+b_2)(b_1+b_2)$, $B_-^{(1a)} = (a_2-b_2)(a_1+a_2+b_2)(b_1+a_2+b_2)$, $B_-^{(1b)} = (a_2-b_2)(a_1+2b_2)(b_1+2b_2)$.

The resulting equations contain interesting symmetries which makes the interpretation easy. We observe that they split into three terms. For $T > 0$ the first term captures the NMDA influence only, while for $T < 0$ it captures the influence of only the BP-spike (apart from scaling factors). Mixed influences arise from the second and third terms which scale with the peak current amplitude $\bar{i}_{total}$ of the BP-spike.

## 3 Results

While the properties of mature NMDA channels are captured by the parameters given for Eq. 2 and remain fairly constant, BP-spikes change their shapes along the dendrite. Thus,

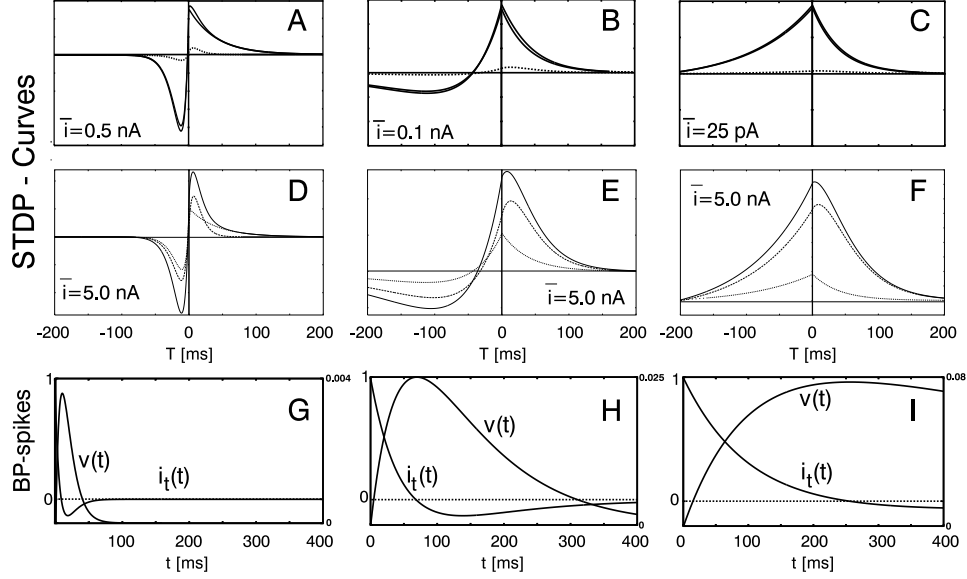

Figure 2: (A-F) STDP-curves obtained from Eqs. 22, 25 and corresponding normalised BP-spikes (G-I, $\bar{i}_{total} = 1$, left y-axis: current, right y-axis: integrated potential). Panels A-C were obtained with different peak currents $\bar{i}_{total} = 0.5\ nA, 0.1nA$ and $25pA$. These currents cause peak voltages of $40mV, 50mV$ and $40mV$ respectively. Panels D-F were all simulated with a peak current of $\bar{i}_{total} = 5.0\ nA$. This current is unrealistic, however, it is chosen for illustrative purposes to show the different contributions to the learning curve (the dashed lines for $G^{(0)}$ and the dotted lines for $G^{(1a,b)}$ and the solid lines for the sum of the two contributions). Time constants for the BP-spikes were: (A,D,G) $a_2^{-1} = \tau_a = 0.0095\ ms, b_2^{-1} = \tau_b = 0.01\ ms$ (B,E,H) $\tau_a = 0.05\ ms, \tau_b = 0.1\ ms$ (C,F,I) $\tau_a = 0.1\ ms, \tau_b = 1.0\ ms$.

we kept the NMDA properties unchanged and varied the time constants of the BP-spikes as well as the current amplitude to simulate this effect. Fig. 2 shows STDP curves (solid lines, A-F) and the corresponding BP-spikes (G-I). The contributions of the different terms to the STDP curves are also shown (first term, dashed, as well as second and third term scaled with their fore-factor, dotted). All curves have arbitrary units. As expected we find that the first term dominates for small (realistic) currents (top panels), while the second and third terms dominate for higher currents (middle panels). Furthermore, we find that long BP-spikes will lead to plain Hebbian learning, where only LTP but no LTD is observed (B,C,E,F).

## 4  Discussion

We believe that two of our findings could be of longer lasting relevance for the understanding of synaptic learning, provided they withstand physiological scrutinising: 1) The shape of the weight change curves heavily relies on the shape of the backpropagating spike. 2) STDP can turn into plain Hebbian learning if the postsynaptic depolarisation (i.e., the BP-spike) rises shallow.

Physiological studies suggest that weight change curves can indeed have a widely varying shape (reviewed in [17]). In this study we argue that in particular the shape of the back-

propagating spike influences the shape of the weight change curve. In fact the dendrites can be seen as active filters which change the shape of backpropagating spikes during their journey to the distal parts of the dendrite [18]. In particular, the decay time of the BP spike is increased in the distal parts of the dendrite [15]. The different decay times determine if we get pure symmetric Hebbian learning or STDP (see Fig. 2). Thus, the theoretical result would suggest temporal symmetric Hebbian learning in the distal dendrites and STDP in the proximal dendrites. From a computational perspective this would mean that the distal dendrites perform principle component analysis [19] and the proximal dendrites temporal sequence learning [20].

Now, our model has to be compared to other models of STDP. We can count our model to the "state variable models". Such models can either adopt a rather descriptive approach [21], where appropriate functions are being fit to the measured weight change curves. Others are closer to the kinetic models in trying to fit phenomenological kinetic equations [7, 22, 23, 9]. Those models establish a more realistic relation between calcium concentration and membrane potential. The calcium concentration seems to be a low-pass filtered version of the membrane potential [24]. Such a low pass filter $h_{low}$ could be added to the learning rule Eq. 3 resulting in: $d\rho/dt = g(t)h_{low}(t) * v'(t)$.

The approaches of [9] as well as of Karmarkar and co-workers [23] are closely related to our model. Both models investigate the effects of different calcium concentration *levels* by assuming certain (e.g. exponential) functional characteristics to govern its changes. This allows them to address the question of how different calcium levels will lead to LTD or LTP [25]. Both model-types [9, 23, 8] were designed to produce a zero-crossing (transition between LTD and LTP) at $T = 0$. The differential Hebbian rule employed by us leads to the observed results as the consequence of the fact that the derivative of any generic unimodal signal will lead to a bimodal curve. We utilise the derivative of the unimodal membrane potential to obtain a bimodal weight change curve. The derivative of the membrane potential is proportional to the charge transfer $\frac{dq_t}{dt} = i_t$ across the (post-synaptic) membrane (see Eq. 7). There is wide ranging support that synaptic plasticity is strongly dominated by calcium transfer through NMDA channels [26, 27, 6]. Thus it seems reasonable to assume that a part of $\frac{dQ}{dt}$ represents calcium flow through the NMDA channel.

## Footnotes

[1]Note that in spines, however, synaptic input can lead to large changes in the postsynaptic potential. In such cases $g(t)$ contributes substantially to $v(t)$.

[2]We use lower-case letters for functions in the time-domain and upper-case letters for their equivalent in the Laplace domain.

# References

[1] D. O. Hebb. *The organization of behavior: A neurophychological study*. Wiley-Interscience, New York, 1949.

[2] H Markram, J Lübke, M Frotscher, and B Sakman. Regulation of synaptic efficacy by coincidence of postsynaptic aps and epsps. *Science*, 275:213–215, 1997.

[3] J. C. Magee and D. Johnston. A synaptically controlled, associative signal for Hebbian plasticity in hippocampal neurons. *Science*, 275:209–213, 1997.

[4] Daniel Johnston, Brian Christie, Andreas Frick, Richard Gray, Dax A. Hoffmann, Lalania K. Schexnayder, Shigeo Watanabe, and Li-Lian Yuan. Active dendrites, potassium channels and synaptic plasticity. *Phil. Trans. R. Soc. Lond. B*, 358:667–674, 2003.

[5] D. J. Linden. The return of the spike: Postsynaptic action potentials and the induction of LTP and LTD. *Neuron*, 22:661–666, 1999.

[6] R. C. Malenka and R. A. Nicoll. Long-term potentiation — a decade of progress? *Science*, 285:1870–1874, 1999.

[7] W. Senn, H. Markram, and M. Tsodyks. An algorithm for modifying neurotransmitter release probability based on pre-and postsynaptic spike timing. *Neural Comp.*, 13:35–67, 2000.

[8] U. R. Karmarkar, M. T. Najarian, and D. V. Buonomano. Mechanisms and significance of spike-timing dependent plasticity. *Biol. Cybern.*, 87:373–382, 2002.

[9] H. Z. Shouval, M. F. Bear, and L. N. Cooper. A unified model of NMDA receptor-dependent bidirectional synaptic plasticity. *Proc. Natl. Acad. Sci. (USA)*, 99(16):10831–10836, 2002.

[10] G. Q. Bi. Spatiotemporal specificity of synaptic plasticity: cellular rules and mechanisms. *Biol. Cybern.*, 87:319–332, 2002.

[11] Patrick D. Roberts. Temporally asymmetric learning rules: I. Differential Hebbian Learning. *Journal of Computational Neuroscience*, 7(3):235–246, 1999.

[12] Richard Kempter, Wulfram Gerstner, and J. Leo van Hemmen. Hebbian learning and spiking neurons. *Physical Review E*, 59:4498–4514, 1999.

[13] R.S. Sutton and A.G. Barto. Towards a modern theory of adaptive networks: Expectation and prediction. *Psychological Review*, 88:135–170, 1981.

[14] C. Koch. *Biophysics of Computation*. Oxford University Press, 1999.

[15] Greg Stuart, Nelson Spruston, Bert Sakmann, and Michael Häusser. Action potential initiation and backpropagation in neurons of the mammalian cns. *Trends Neurosci.*, 20(3):125–131, 1997.

[16] John L. Stewart. *Fundamentals of Signal Theory*. Mc Graw-Hill, New York, 1960.

[17] P. D. Roberts and C. C. Bell. Spike timing dependent synaptic plasticity in biological systems. *Biol. Cybern.*, 87:392–403, 2002.

[18] Nace L. Golding, William L. Kath, and Nelson Spruston. Dichotomy of action potential backpropagation in ca1 pyramidal neuron dendrites. *J Neurophysiol*, 86:2998–3009, 2001.

[19] E. Oja. A simplified neuron model as a principal component analyzer. *J Math Biol*, 15(3):267–273, 1982.

[20] Bernd Porr and Florentin Wörgötter. Isotropic Sequence Order learning. *Neural Computation*, 15:831–864, 2003.

[21] H. D. I. Abarbanel, R. Huerta, and M. I. Rabinovich. Dynamical model of long-term synaptic plasticity. *Proc. Natl. Acad. Sci. (USA)*, 99(15):10132–10137, 2002.

[22] G. C. Castellani, E. M. Quinlan, L. N. Cooper, and H. Z. Shouval. A biophysical model of bidirectional synaptic plasticity: Dependence on AMPA and NMDA receptors. *Proc. Natl. Acad. Sci. (USA)*, 98(22):12772–12777, 2001.

[23] U. R. Karmarkar and D. V. Buonomano. A model of spike-timing dependent plasticity: One or two coincidence detectors? *J. Neurophysiol.*, 88:507–513, 2002.

[24] G. Stuart, J. Schiller, and B. Sakmann. Action potential initiation and propagation in rat neocortical pyramidal neurons. *J Physiol*, 505:617–632, 1997.

[25] M. Nishiyama, K. Hong, K. Mikoshiba, M. Poo, and K. Kato. Calcium stores regulate the polarity and input specificity of synaptic modification. *Nature*, 408:584–588, 2000.

[26] J. Schiller, Y. Schiller, and D. E. Clapham. Amplification of calcium influx into dendritic spines during associative pre- and postsynaptic activation: The role of direct calcium influx through the NMDA receptor. *Nat. Neurosci.*, 1:114–118, 1998.

[27] R. Yuste, A. Majewska, S. S. Cash, and W. Denk. Mechanisms of calcium influx into hippocampal spines: heterogeneity among spines, coincidence detection by NMDA receptors, and optical quantal analysis. *J. Neurosci.*, 19:1976–1987, 1999.
